# Latent Coincidence Analysis: A Hidden Variable Model for Distance Metric Learning

**Matthew Der and Lawrence K. Saul**
Department of Computer Science and Engineering
University of California, San Diego
La Jolla, CA 92093
{mfder,saul}@cs.ucsd.edu

## Abstract

We describe a latent variable model for supervised dimensionality reduction and distance metric learning. The model discovers linear projections of high dimensional data that shrink the distance between similarly labeled inputs and expand the distance between differently labeled ones. The model's continuous latent variables locate pairs of examples in a latent space of lower dimensionality. The model differs significantly from classical factor analysis in that the posterior distribution over these latent variables is *not* always multivariate Gaussian. Nevertheless we show that inference is completely tractable and derive an Expectation-Maximization (EM) algorithm for parameter estimation. We also compare the model to other approaches in distance metric learning. The model's main advantage is its simplicity: at each iteration of the EM algorithm, the distance metric is re-estimated by solving an unconstrained least-squares problem. Experiments show that these simple updates are highly effective.

## 1  Introduction

In this paper we propose a simple but new model to learn informative linear projections of multivariate data. Our approach is rooted in the tradition of latent variable modeling, a popular methodology for discovering low dimensional structure in high dimensional data. Two well-known examples of latent variable models are factor analyzers (FAs), which recover subspaces of high variance [1], and Gaussian mixture models (GMMs), which reveal clusters of high density [2]. Here we describe a model that we call latent coincidence analysis (LCA). The goal of LCA is to discover a latent space in which metric distances reflect meaningful notions of similarity and difference.

We apply LCA to two problems in distance metric learning, where the goal is to improve the performance of a classifier—typically, a $k$-nearest neighbor (kNN) classifier [3]—by a linear transformation of its input space. Several previous methods have been proposed for this problem, including neighborhood component analysis (NCA) [4], large margin neighbor neighbor classification (LMNN) [5], and information-theoretic metric learning (ITML) [6]. These methods—all of them successful, all of them addressing the same problem—beg the obvious question: *why yet another?*

One answer is suggested by the different lineages of previous approaches. NCA was conceived as a supervised counterpart to stochastic neighborhood embedding [7], an unsupervised method for dimensionality reduction. LMNN was conceived as a kNN variant of support vector machines [8]. ITML evolved from earlier work in Bregman optimizations—that of minimizing the LogDet divergence subject to linear constraints [9]. Perhaps it is due to

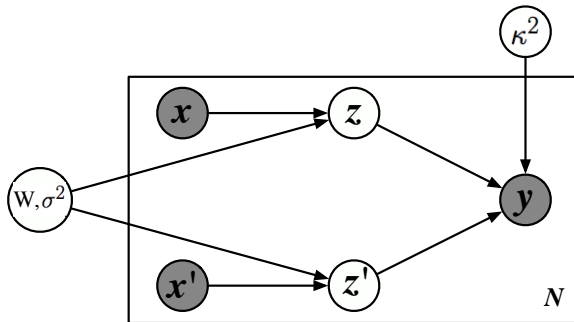

Figure 1: Bayesian network for latent coincidence analysis. The inputs $\mathbf{x}, \mathbf{x}' \in \Re^d$ are mapped into Gaussian latent variables $\mathbf{z}, \mathbf{z}' \in \Re^p$ whose statistics are parameterized by the linear transformation $\mathbf{W} \in \Re^{p \times d}$ and noise level $\sigma$. Coincidence in the latent space at length scale $\kappa$ is detected by the binary variable $y \in \{0, 1\}$. Observed nodes are shaded.

these different lineages that none of these methods completely dominates the others. They all offer improvements in kNN classification, yet arguably their larger worth stems from the related work they have inspired in other areas of pattern recognition. Distance metric learning is a fundamental problem, and the more solutions we have, the better equipped we are to solve its myriad variations.

It is in this spirit that we revisit the problem of distance metric learning in the venerable tradition of latent variable modeling. We believe that LCA, like factor analysis and Gaussian mixture modeling, is the simplest latent variable model that can be imagined for its purpose. In particular, the inference in LCA (though not purely Gaussian) is tractable, and the distance metric is re-estimated at each iteration of its EM algorithm by a simple least-squares update. This update has stronger guarantees of convergence than the gradient-based methods in NCA; it also sidesteps the large number of linear inequality constraints that appear in the optimizations for LMNN and ITML. For all these reasons, we believe that LCA deserves to be widely known.

## 2 Model

We begin by describing the probabilistic model for LCA. Fig. 1 shows the model's representation as a Bayesian network. There are three observed variables: the inputs $\mathbf{x}, \mathbf{x}' \in \Re^d$, which we always imagine to be observed in pairs, and the binary label $y \in \{0, 1\}$, which indicates if the inputs map (or are desired to be mapped) to nearby locations in a latent space of equal or reduced dimensionality $p \leq d$. These locations are in turn represented by the Gaussian latent variables $\mathbf{z}, \mathbf{z}' \in \Re^p$.

Each node in the Bayesian network is conditionally dependent on its parents. The conditional distributions $P(\mathbf{z}|\mathbf{x})$ and $P(\mathbf{z}'|\mathbf{x}')$ are parameterized by a linear transformation $\mathbf{W} \in \Re^{p \times d}$ (from the input space to the latent space) and a noise level $\sigma^2$. They take the simple Gaussian form:

$$P(\mathbf{z}|\mathbf{x}) = (2\pi\sigma^2)^{-p/2} e^{-\frac{1}{2\sigma^2} \|\mathbf{z} - \mathbf{W}\mathbf{x}\|^2}, \tag{1}$$

$$P(\mathbf{z}'|\mathbf{x}') = (2\pi\sigma^2)^{-p/2} e^{-\frac{1}{2\sigma^2} \|\mathbf{z}' - \mathbf{W}\mathbf{x}'\|^2}. \tag{2}$$

Finally, the binary label $y \in \{0, 1\}$ is used to detect the coincidence of the variables $\mathbf{z}, \mathbf{z}'$ in the latent space. In particular, $y$ follows a Bernoulli distribution with mean value:

$$P(y{=}1|\mathbf{z}, \mathbf{z}') = e^{-\frac{1}{2\kappa^2} \|\mathbf{z} - \mathbf{z}'\|^2}. \tag{3}$$

Eq. (3) states that $y{=}1$ with certainty if $\mathbf{z}$ and $\mathbf{z}'$ coincide at the exact same point in the latent space; otherwise, the probability in eq. (3) falls off exponentially with their squared distance. The length scale $\kappa$ in eq. (3) governs the rate of this exponential decay.

## 2.1 Inference

Inference in this model requires averaging over the Gaussian latent variables $\mathbf{z}, \mathbf{z}'$. The required integrals take the form of simple Gaussian convolutions. For example:

$$P(y{=}1|\mathbf{x}, \mathbf{x}') \;\; = \;\; \int \mathbf{dz}\,\mathbf{dz}'\, P(y{=}1|\mathbf{z}, \mathbf{z}')\, P(\mathbf{z}|\mathbf{x})\, P(\mathbf{z}'|\mathbf{x}') \tag{4}$$

$$= \;\; \left( \frac{\kappa^2}{\kappa^2 + 2\sigma^2} \right)^{p/2} \exp\left( -\frac{\|\mathbf{W}(\mathbf{x}-\mathbf{x}')\|^2}{2\,(\kappa^2+2\sigma^2)} \right). \tag{5}$$

Note that this marginal probability is invariant to uniform re-scalings of the model parameters $\mathbf{W}$, $\sigma$, and $\kappa$; we will return to this observation later. For inputs $(\mathbf{x}, \mathbf{x}')$, we denote the relative likelihood, or *odds*, of the event $y{=}1$ by

$$\nu(\mathbf{x}, \mathbf{x}') \;\; = \;\; \frac{P(y{=}1|\mathbf{x}, \mathbf{x}')}{P(y{=}0|\mathbf{x}, \mathbf{x}')}. \tag{6}$$

As we shall see, the odds appear in the calculations for many useful forms of inference. Note that the odds $\nu(\mathbf{x}, \mathbf{x}')$ has a complicated nonlinear dependence on the inputs $(\mathbf{x}, \mathbf{x}')$; the numerator in eq. (6) is Gaussian, but the denominator (equal to one minus the numerator) is not.

Of special importance for learning (as discussed in section 2.2) are the statistics of the posterior distribution $P(\mathbf{z}, \mathbf{z}'|\mathbf{x}, \mathbf{x}', y)$. We obtain this distribution using Bayes rule:

$$P(\mathbf{z}, \mathbf{z}'|\mathbf{x}, \mathbf{x}', y) \;\; = \;\; \frac{P(y|\mathbf{z}, \mathbf{z}')\, P(\mathbf{z}|\mathbf{x})\, P(\mathbf{z}'|\mathbf{x}')}{P(y|\mathbf{x}, \mathbf{x}')} \;\;. \tag{7}$$

We note that the prior distribution $P(\mathbf{z}, \mathbf{z}'|\mathbf{x}, \mathbf{x}')$ is multivariate Gaussian, as is the posterior distribution $P(\mathbf{z}, \mathbf{z}'|\mathbf{x}, \mathbf{x}', y = 1)$ for positively labeled pairs of examples. However, this is *not* true of the posterior distribution $P(\mathbf{z}, \mathbf{z}'|\mathbf{x}, \mathbf{x}', y = 0)$ for negatively labeled pairs. In this respect, the model differs from classical factor analysis and other canonical models with Gaussian latent variables (e.g., Kalman filters).

Despite the above wrinkle, it remains straightforward to compute the low-order moments of the distribution in eq. (7) for both positively ($y{=}1$) and negatively ($y{=}0$) labeled pairs[1] of examples. In particular, for the posterior means, we obtain:

$$\mathrm{E}[\mathbf{z}|\mathbf{x}, \mathbf{x}', y{=}0] \;\; = \;\; \mathbf{W}\left[ \mathbf{x} - \left( \frac{\nu\sigma^2}{\kappa^2 + 2\sigma^2} \right)(\mathbf{x}'-\mathbf{x}) \right], \tag{8}$$

$$\mathrm{E}[\mathbf{z}|\mathbf{x}, \mathbf{x}', y{=}1] \;\; = \;\; \mathbf{W}\left[ \mathbf{x} + \left( \frac{\sigma^2}{\kappa^2 + 2\sigma^2} \right)(\mathbf{x}'-\mathbf{x}) \right], \tag{9}$$

where the coefficient $\nu$ in eq. (8) is shorthand for the odds $\nu(\mathbf{x}, \mathbf{x}')$ in eq. (6). Note how the posterior means $\mathrm{E}[\mathbf{z}|\mathbf{x}, \mathbf{x}', y]$ in eqs. (8–9) differ from the prior mean

$$\mathrm{E}[\mathbf{z}|\mathbf{x}, \mathbf{x}'] = \mathbf{W}\mathbf{x}. \tag{10}$$

Analogous results hold for the prior and posterior means of the latent variable $\mathbf{z}'$. Intuitively, these calculations show that the expected values of $\mathbf{z}$ and $\mathbf{z}'$ move toward each other if the observed label indicates a coincidence ($y{=}1$) and away from each other if not ($y{=}0$).

For learning it is also necessary to compute second-order statistics of the posterior distribution. For the posterior variances, straightforward calculations give:

$$\mathrm{E}\left[ \|\mathbf{z} - \bar{\mathbf{z}}\|^2 \,\bigg|\, \mathbf{x}, \mathbf{x}', y{=}0 \right] \;\; = \;\; p\sigma^2\left[ 1 + \frac{\nu\sigma^2}{\kappa^2 + 2\sigma^2} \right], \tag{11}$$

$$\mathrm{E}\left[ \|\mathbf{z} - \bar{\mathbf{z}}\|^2 \,\bigg|\, \mathbf{x}, \mathbf{x}', y{=}1 \right] \;\; = \;\; p\sigma^2\left[ 1 - \frac{\sigma^2}{\kappa^2 + 2\sigma^2} \right], \tag{12}$$

where $\bar{\mathbf{z}}$ in these expressions denotes the posterior means in eqs. (8–9), and again the coefficient $\nu$ is shorthand for the odds $\nu(\mathbf{x}, \mathbf{x}')$ in eq. (6). Note how the posterior variances in eqs. (11–12) differ from the prior variance

$$\mathrm{E}\left[\|\mathbf{z} - \mathbf{W}\mathbf{x}\|^2 \,\bigg|\, \mathbf{x}, \mathbf{x}'\right] = p\sigma^2. \tag{13}$$

Intuitively, we see that the posterior variance shrinks if the observed label indicates a coincidence $(y = 1)$ and grows if not $(y = 0)$. The expressions for the posterior variance of the latent variable $\mathbf{z}'$ are identical due to the model's symmetry.

## 2.2 Learning

Next we consider how to learn the linear projection $\mathbf{W}$, the noise level $\sigma^2$, and the length scale $\kappa^2$ from data. We assume that the data comes in the form of paired inputs $\mathbf{x}, \mathbf{x}'$, together with binary judgments $y \in \{0, 1\}$ of similarity or difference. In particular, from a training set $\{(\mathbf{x}_i, \mathbf{x}_i', y_i)\}_{i=1}^N$ of $N$ such examples, we wish to learn the parameters that maximize the conditional log-likelihood

$$\mathcal{L}(\mathbf{W}, \sigma^2, \kappa^2) = \sum_{i=1}^N \log P(y_i | \mathbf{x}_i, \mathbf{x}_i') \tag{14}$$

of observed coincidences $(y_i = 1)$ and non-coincidences $(y_i = 0)$. We say that the data is *incomplete* or *partially observed* in the sense that the examples do not specify target values for the latent variables $\mathbf{z}, \mathbf{z}'$; instead, such target values must be inferred from the model's posterior distribution.

Given values for the model parameters $\mathbf{W}, \sigma^2$, and $\kappa^2$, we can compute the right hand side of eq. (14) from the result in eq. (5). However, the parameters that maximize eq. (14) cannot be computed in closed form. In the absence of an analytical solution, we avail ourselves of the EM algorithm, an iterative procedure for maximum likelihood estimation in latent variable models [10]. The EM algorithm consists of two steps, an E-step which computes statistics of the posterior distribution in eq. (7), and an M-step which uses these statistics to re-estimate the model parameters. The two steps are iterated until convergence.

Intuitively, the EM algorithm uses the posterior means in eqs. (8–9) to "fill in" the missing values of the latent variables $\mathbf{z}, \mathbf{z}'$. As shorthand, let

$$\begin{align}
\bar{\mathbf{z}}_i &= \mathrm{E}[\mathbf{z} | \mathbf{x}_i, \mathbf{x}_i', y_i], \tag{15} \\
\bar{\mathbf{z}}_i' &= \mathrm{E}[\mathbf{z}' | \mathbf{x}_i, \mathbf{x}_i', y_i] \tag{16}
\end{align}$$

denote these posterior means for the $i$th example in the training set, as computed from the results in eq. (8–9). The M-step of the EM algorithm updates the linear transformation $\mathbf{W}$ by minimizing the sum of squared errors

$$\mathcal{E}(\mathbf{W}) = \frac{1}{2} \sum_{i=1}^N \left[ \|\bar{\mathbf{z}}_i - \mathbf{W}\mathbf{x}_i\|^2 + \|\bar{\mathbf{z}}_i' - \mathbf{W}\mathbf{x}_i'\|^2 \right], \tag{17}$$

where the expected values $\bar{\mathbf{z}}_i, \bar{\mathbf{z}}_i'$ are computed with respect to the *current* model parameters (and thus treated as constants in the above minimization). Minimizing the sum of squared errors in eq. (17) gives the update rule:

$$\mathbf{W} \leftarrow \left[\sum_{i=1}^N \left(\bar{\mathbf{z}}_i \mathbf{x}_i^\top + \bar{\mathbf{z}}_i' \mathbf{x}_i'^\top\right)\right]\left[\sum_{i=1}^N \left(\mathbf{x}_i \mathbf{x}_i^\top + \mathbf{x}_i' \mathbf{x}_i'^\top\right)\right]^{-1}, \tag{18}$$

where the product in eq. (18) is understood as a vector-matrix multiplication. The EM update for the noise level $\sigma^2$ takes an equally simple form. As shorthand, let

$$\varepsilon_i^2 = \mathrm{E}\left[\|\mathbf{z} - \bar{\mathbf{z}}\|^2 \,\bigg|\, \mathbf{x}_i, \mathbf{x}_i', y_i\right] \tag{19}$$

|  | MNIST | BBC | Classic4 | Isolet | Letters | Seg | Bal | Iris |
|---|---|---|---|---|---|---|---|---|
| # train | 60000 | 1558 | 4257 | 6238 | 14000 | 210 | 438 | 105 |
| # test | 10000 | 667 | 1419 | 1559 | 6000 | 2100 | 187 | 45 |
| # classes | 10 | 5 | 4 | 26 | 26 | 7 | 3 | 3 |
| # features $(D)$ | 784 | 9635 | 5896 | 617 | 16 | 19 | 4 | 4 |
| # inputs $(d)$ | 164 | 200 | 200 | 172 | 16 | 19 | 4 | 4 |
| LCA dim $(p)$ | 40 | 4 | 32 | 40 | 16 | 18 | 4 | 4 |
| Euclidean | 2.83 | 30.10 | 7.89 | 8.98 | 4.73 | 13.71 | 18.82 | 5.33 |
| PCA | 2.12 | 11.90 | 9.74 | 8.60 | 4.75 | 13.71 | 18.07 | 5.33 |
| LMNN | 1.34 | 3.40 | 3.19 | 3.40 | 2.58 | 8.57 | 8.98 | 5.33 |
| LCA | 1.61 | 3.41 | 3.54 | 3.72 | 2.93 | 8.57 | 4.06 | 2.22 |

Table 1: Summary of data sets for multiway classification. For efficiency we projected data sets of high dimensionality $D$ down to their leading $d$ principal components. MNIST [11] is a data set of handwritten digits; we deslanted the images to reduce variability. BBC [12] and Classic4 [13] are text corpora with labeled topics. The last five data sets are from the UCI repository [14]. The bottom four rows compare test error percentage using 3-NN classification. For data sets without dedicated test sets, we averaged across multiple random 70/30 splits.

denote the posterior variance of $\mathbf{z}$ for the $i$th example in the training set, as computed from the results in eqs. (11–12). Then the EM update for the noise level is given by:

$$\sigma^2 \ \leftarrow \ \frac{1}{pN} \left[ \min_{\mathbf{W}} \mathcal{E}(\mathbf{W}) + \sum_{i=1}^{N} \varepsilon_i^2 \right]. \tag{20}$$

The minimum of $\mathcal{E}(\mathbf{W})$ in this update is computed by substituting the right hand side of eq. (18) into eq. (17).

The EM updates for $\mathbf{W}$ and $\sigma^2$ have the desirable property that they converge monotonically to a stationary point of the log-likelihood: that is, at each iteration, *they are guaranteed to increase the right hand side of eq. (14)* except at points in the parameter space with vanishing gradient. A full derivation of the EM algorithm is omitted for brevity.

We have already noted that the log-likelihood in eq. (14) is invariant to uniform rescaling of the model parameters $\mathbf{W}$, $\sigma$, and $\kappa$. Thus without loss of generality we can set $\kappa^2 = 1$ in the simplest setting of the model, as described above. It does become necessary to estimate the parameter $\kappa^2$, however, in slightly extended formulations of the model, as we consider in section 3.2. Unlike the parameters $\mathbf{W}$ and $\sigma^2$, the parameter $\kappa^2$ does not have a simple update rule for its re-estimation by EM. When necessary, however, this parameter can be re-estimated by a simple line search. This approach also preserves the property of monotonic convergence.

## 3 Applications

We explore two applications of LCA in which its linear transformation is used to preprocess the data for different models of multiway classification. We assume that the original data consists of labeled examples $\{(\mathbf{x}_i, c_i)\}$ of inputs $\mathbf{x}_i \in \Re^d$ and their class labels $c_i \in \{1, 2, \ldots, c\}$. For each application, we show how to instantiate LCA by creating a particular data set of labeled pairs, where the labels indicate whether the examples in each pair should be mapped closer together $(y = 1)$ or farther apart $(y = 0)$ in LCA's latent space of dimensionality $p \leq d$. In the first application, we use LCA to improve a parametric model of classification; in the second application, a nonparametric one. The data sets in our experiments are summarized in Table 1.

### 3.1 Gaussian mixture modeling

Gaussian mixture models (GMMs) offer perhaps the simplest parametric model of multiway classification. In the most straightforward application of GMMs, the labeled examples in

each class $c$ are modeled by a single multivariate Gaussian distribution with mean $\boldsymbol{\mu}_c$ and covariance matrix $\boldsymbol{\Sigma}_c$. Classification is also simple: for each unlabeled example, we use Bayes rule to compute the class with the highest posterior probability.

Even in these simplest of GMMs, however, challenges arise when the data is very high dimensional. In this case, it may be prohibitively expensive to estimate or store the covariance matrix for each class of the data. In this case two simple options are: (i) to reduce the input's dimensionality using principal component analysis (PCA) or linear discriminant analysis (LDA) [15], or (ii) to model each multivariate Gaussian distribution using factor analysis. In the latter, we learn distributions of the form:

$$P(\mathbf{x}|c) \;\sim\; \mathcal{N}(\boldsymbol{\mu}_c, \boldsymbol{\Psi}_c + \boldsymbol{\Lambda}_c \boldsymbol{\Lambda}_c^\top) \tag{21}$$

where the diagonal matrix $\boldsymbol{\Psi}_c \in \Re^{d \times d}$ and loading matrix $\boldsymbol{\Lambda}_c \in \Re^{d \times p}$ are the model parameters of the factor analyzer belonging to the $c^{\text{th}}$ class. Factor analysis can be formulated as a latent variable model, and its parameters estimated by an EM algorithm [1].

GMMs are generative models trained by maximum likelihood estimation. In this section, we explore how LCA may yield classifiers of similar form but higher accuracy. To do so, we learn one model of LCA for each class of the data. In particular, we use LCA to project each example $\mathbf{x}_i$ into a lower dimensional space where we hope for two properties: (i) that it is *closer* to the mean of projected examples from the same class $y_i$, and (ii) that it is *farther* from the mean of projected examples from other classes $c \neq y_i$.

More specifically, we instantiate the model of LCA for each class as follows. Let $\boldsymbol{\mu}_c \in \Re^d$ denote the mean of the labeled examples in class $c$. Then we create a training set of labeled pairs $\{\boldsymbol{\mu}_c, \mathbf{x}_i, y_{ic}\}$ over *all* examples $\mathbf{x}_i$ where $y_{ic} = 1$ if $y_i = c$ and $y_{ic} = 0$ if $y_i \neq c$. From this training set, we use the EM algorithm in section 2.2 to learn a (class-specific) linear projection $\mathbf{W}_c$ and variance $\sigma_c$. Finally, to classify an unlabeled example $\mathbf{x}$, we compute the probabilities:

$$P(y_c = 1|\mathbf{x}) \;=\; \left(\frac{1}{1 + 2\sigma_c^2}\right)^{p/2} \exp\left\{-\frac{\|\mathbf{W}_c(\mathbf{x} - \boldsymbol{\mu}_c)\|^2}{2\,(1 + 2\sigma_c^2)}\right\}. \tag{22}$$

We label the example $\mathbf{x}$ by the class $c$ that maximizes the probability in eq. (22). As we shall see, this decision rule for LCA often makes different predictions than Bayes rule in maximum likelihood GMMs. Conveniently, we can train the LCA models for different classes in parallel.

We evaluated the performance of LCA in this setting on the first four data sets in Table 1. Over a range of reduced dimensionalities $p < d$, we compared the classification accuracy of three approaches: (i) GMMs with full covariance matrices after projecting the data down to $p$ dimensions with PCA or LDA, (ii) GMMs with $p$-dimensional factor analyzers, and (iii) $p$-dimensional models of LCA. Fig. 2 shows that LCA generally outperforms these other methods; also, its largest gains occur in the regime of very aggressive dimensionality reduction $p \ll d$. To highlight the results in this regime, Fig. 3 contrasts the $p = 2$ dimensional representations of the data discovered by PCA and LCA. Here it is visually apparent that LCA leads to much better separation of the examples in different classes.

## 3.2   Distance metric learning

We can also apply LCA to learn a distance metric that improves kNN classification [4, 5, 6]. Our approach draws heavily on the ideas of LMNN [5], though differs in its execution. In LMNN, each training example has $k$ target neighbors, typically chosen as the $k$ nearest neighbors in Euclidean space with the same class label. LMNN learns a metric to shrink the distances between examples and target neighbors while preserving (or increasing) the distances between examples from different classes. Errors in kNN classification tend to occur when differently labeled examples are closer together than pairs of target neighbors. Thus LMNN seeks to minimize the number of differently labeled examples that invade the perimeters established by target neighbors. These examples are known as *impostors*.

In LCA, we can view the matrix $\mathbf{W}^\top \mathbf{W}$ as a Mahalanobis distance metric for kNN classification. The starting point of LCA is to create a training set of pairs of examples. Among

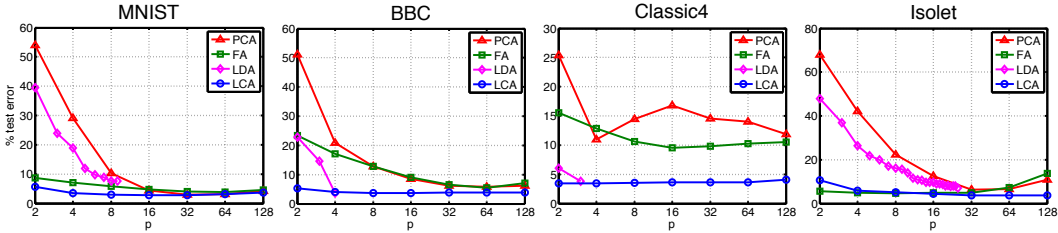

Figure 2: Comparison of dimensionality reduction by principal components analysis (PCA), factor analysis (FA), linear discriminant analysis (LDA), and latent coincidence analysis (LCA). The plots show test set error versus dimensionality $p$.

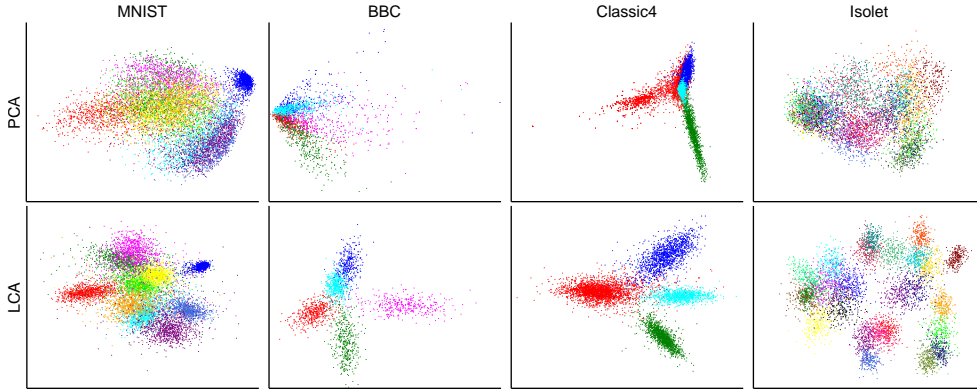

Figure 3: Comparison of two-dimensional ($p=2$) representations of data discovered by PCA and LCA. The examples are color-coded by class label.

these pairs, we wish the similarly labeled examples to coincide ($y=1$) and the differently labeled examples to diverge ($y=0$). For the former, it is natural to choose all pairs of examples and their target neighbors. For the latter, it is natural to choose all pairs of differently labeled examples. Concretely, if there are $c$ classes, each with $m$ examples, then this approach creates a training set of $ckm$ pairs of similarly labeled examples (with $y=1$) and $c(c-1)m^2$ pairs of differently labeled examples (with $y=0$).

Unfortunately it is clear that this approach does not scale well with the number of examples. We therefore adopt two pruning strategies in our implementation of LCA. First, we do not include training examples without impostors. Second, among the pairs of differently labeled examples, we only include each example with its current or previous impostors. A complication of this approach is that every few iterations we must check to see if any example has new impostors. If so, we add the example, its target neighbors, and impostors into the training set. This strategy was used in all the experiments described below. Our short-cuts are similar in spirit to the optimizations in LMNN [5] as well as more general cutting plane strategies of constrained optimization [16].

The use of LCA for kNN classification also benefits from a slight but crucial extension of the model in Fig. 1. Recall that the parameter $\kappa^2$ determines the *length scale* at which projected examples are judged to coincide in the latent space. For kNN classification, we extend the model in Fig. 1 to learn a local parameter $\kappa^2$ for each input in the training set. These local parameters $\kappa^2$ are needed to account for the fact that different inputs may reside at very different distances from their target neighbors. In the graphical model of Fig. 1, this extension amounts to drawing an *additional* plate that encloses the parameter $\kappa^2$ and the model's random variables, but not the parameters $\mathbf{W}$ and $\sigma^2$.

Note that the $\sigma^2$ and $\kappa^2$ parameters of LCA, though important to estimate, are not ultimately used for kNN classification. In particular, after a model is trained, we simply

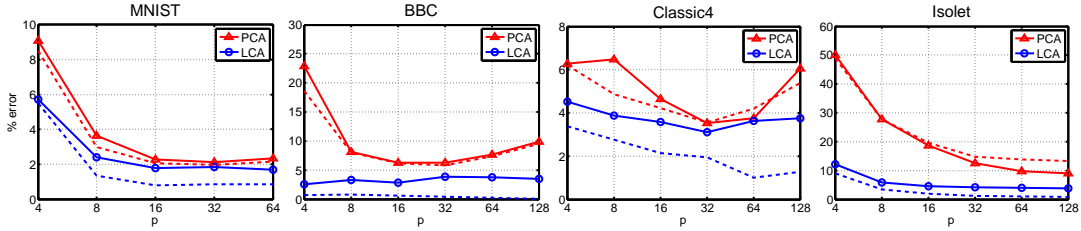

Figure 4: Comparison of dimensionality reduction by principal components analysis (PCA) and latent coincidence analysis (LCA). The plots show kNN classification error (training dotted, test solid) versus dimensionality $p$.

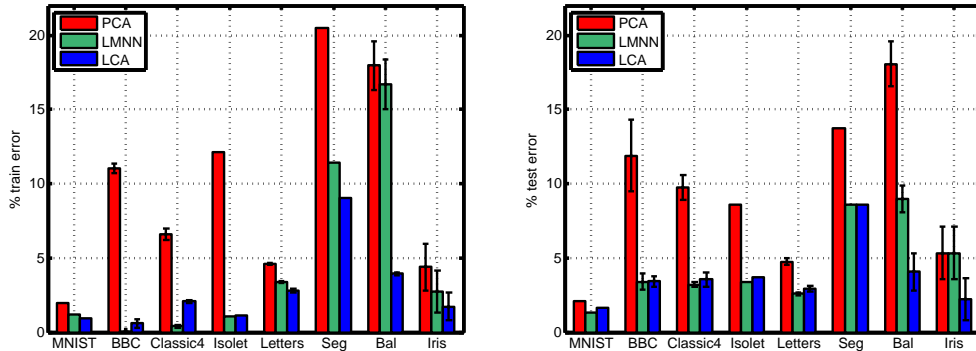

Figure 5: Comparison of kNN classification by PCA, LMNN, and LCA. We set $k=3$ for all experiments. Training error is computed using leave-one-out cross-validation. The values of $p$ used for LCA are given in Table 1.

perform kNN classification using the Mahalanobis distance metric parameterized by the linear transformation $\mathbf{W}$.

For comparison, we measure kNN classification accuracy using Euclidean distance, PCA, and LMNN. We report all three along with LCA in Table 1, but we focus on PCA in Fig. 4 to illustrate the effect of dimensionality reduction. LCA consistently outperforms PCA across all dimensionalities. Additionally, we hold out a validation set to search for an optimal dimensionality $p$. In Fig. 5, we compare LCA to PCA and LMNN. Again, LCA is clearly superior to PCA and generally achieves comparable performance to LMNN. Advantageously, we often obtain our best result with LCA using a lower dimensionality $p < d$.

## 4    Discussion

In this paper we have introduced Latent Coincidence Analysis (LCA), a latent variable model for learning linear projections that map similar inputs closer together and different inputs farther apart. Inference in LCA is entirely tractable, and we use an EM algorithm to learn maximum likelihood estimates of the model parameters. Our approach values simplicity, but not at the expense of efficacy. On problems in mixture modeling and distance metric learning tasks, LCA performs competitively across a range of reduced dimensionalities.

There are many directions for future work. One challenge that we observed was slow convergence of the EM algorithm, an issue that may be ameliorated by the gradient or second-order methods proposed in  [17]. To handle larger data sets, we plan to explore online strategies for distance metric learning [18], possibly based on Bayesian [19] or confidence-weighted updates [20]. Finally, we will explore hybrid strategies between the mixture modeling in section 3.1 and kNN classifiction in section  3.2, where multiple (but not all) examples in each class are used as "anchors" for distance-based classification. All these directions should open the door to implementations on larger scales [21] than we have considered here.

## Footnotes

[1]For the latter, the statistics can be expressed as the *differences* of Gaussian integrals.

# References

[1] D. B. Rubin and D. T. Thayer. EM algorithms for ML factor analysis. *Psychometrika*, 47:69–76, 1982.

[2] G. McLachlan and K. Basford. *Mixture Models: Inference and Applications to Clustering*. Marcel Dekker, 1988.

[3] T. Cover and P. Hart. Nearest neighbor pattern classification. In *IEEE Transactions in Information Theory, IT-13*, pages 21–27, 1967.

[4] J. Goldberger, S. Roweis, G. Hinton, and R. Salakhutdinov. Neighbourhood components analysis. In L. K. Saul, Y. Weiss, and L. Bottou, editors, *Advances in Neural Information Processing Systems 17*, pages 513–520, Cambridge, MA, 2005. MIT Press.

[5] K.Q. Weinberger and L.K. Saul. Distance metric learning for large margin nearest neighbor classification. *The Journal of Machine Learning Research*, 10:207–244, 2009.

[6] J. V. Davis, B. Kulis, P. Jain, S. Sra, and I. S. Dhillon. Information-theoretic metric learning. In *ICML*, pages 209–216, Corvalis, Oregon, USA, June 2007.

[7] G. Hinton and S. Roweis. Stochastic neighbor embedding. In S. Thrun S. Becker and K. Obermayer, editors, *Advances in Neural Information Processing Systems 15*, pages 833–840. MIT Press, Cambridge, MA, 2003.

[8] C. Cortes and V. Vapnik. Support-vector networks. *Machine Learning*, 20:273–297, 1995.

[9] B. Kulis, M. A. Sustik, and I. S. Dhillon. Learning low-rank kernel matrices. In *Proceedings of the Twenty-Third International Conference on Machine Learning (ICML-06)*, 2006.

[10] A. P. Dempster, N. M. Laird, and D. B. Rubin. Maximum likelihood from incomplete data via the EM algorithm. *Journal of the Royal Statistical Society B*, 39:1–37, 1977.

[11] http://yann.lecun.com/exdb/mnist/.

[12] http://mlg.ucd.ie/datasets/bbc.html.

[13] http://www.dataminingresearch.com/index.php/2010/09/classic3-classic4-datasets/.

[14] http://archive.ics.uci.edu/ml/datasets.html.

[15] R A Fisher. The use of multiple measurements in taxonomic problems. *Annals of Eugenics*, 7(2):179–188, 1936.

[16] S. Boyd and L. Vandenberghe. *Convex Optimization*. Cambridge University Press, 2004.

[17] R. Salakhutdinov, S. T. Roweis, and Z. Ghahramani. On the convergence of bound optimization algorithms. In *Proceedings of the Nineteenth Conference on Uncertainty in Artificial Intelligence (UAI-03)*, pages 509–516, 2003.

[18] S. Shalev-Shwartz, Y. Singer, and A. Y. Ng. Online and batch learning of pseudo-metrics. In *Proceedings of the Twenty First International Conference on Machine Learning (ICML-04)*, pages 94–101, Banff, Canada, 2004.

[19] T. Jaakkola and M. Jordan. A variational approach to bayesian logistic regression models and their extensions. In *Proceedings of the Sixth International Workshop on Artificial Intelligence and Statistics*, 1997.

[20] M. Dredze, K. Crammer, and F. Pereira. Confidence-weighted linear classification. In Andrew McCallum and Sam Roweis, editors, *Proceedings of the 25th Annual International Conference on Machine Learning (ICML 2008)*, pages 264–271. Omnipress, 2008.

[21] G. Chechik, U. Shalit, V. Sharma, and S. Bengio. An online algorithm for large scale image similarity learning. In Y. Bengio, D. Schuurmans, J. Lafferty, C. K. I. Williams, and A. Culotta, editors, *Advances in Neural Information Processing Systems 22*, pages 306–314. 2009.

